# One microphone blind dereverberation based on quasi-periodicity of speech signals

**Tomohiro Nakatani, Masato Miyoshi, and Keisuke Kinoshita**
Speech Open Lab., NTT Communication Science Labs., NTT Corporation
2-4, Hikaridai, Seika-cho, Soraku-gun, Kyoto, Japan
{nak,miyo,kinoshita}@cslab.kecl.ntt.co.jp

## Abstract

Speech dereverberation is desirable with a view to achieving, for example, robust speech recognition in the real world. However, it is still a challenging problem, especially when using a single microphone. Although blind equalization techniques have been exploited, they cannot deal with speech signals appropriately because their assumptions are not satisfied by speech signals. We propose a new dereverberation principle based on an inherent property of speech signals, namely quasi-periodicity. The present methods learn the dereverberation filter from a lot of speech data with no prior knowledge of the data, and can achieve high quality speech dereverberation especially when the reverberation time is long.

## 1 Introduction

Although numerous studies have been undertaken on robust automatic speech recognition (ASR) in the real world, long reverberation is still a serious problem that severely degrades the ASR performance [1]. One simple way to overcome this problem is to dereverberate the speech signals prior to ASR, but this is also a challenging problem, especially when using a single microphone. For example, certain blind equalization methods, including independent component analysis (ICA), can estimate the inverse filter of an unknown impulse response convolved with target signals when the signals are statistically independent and identically distributed sequences [2]. However, these methods cannot appropriately deal with speech signals because speech signals have inherent properties, such as periodicity and formant structure, making their sequences statistically dependent. This approach inevitably destroys such essential properties of speech. Another approach that uses the properties of speech has also been proposed [3]. The basic idea involves adaptively detecting time regions in which signal-to-reverberation ratios become small, and attenuating speech signals in those regions. However, the precise separation of the signal and reverberation durations is difficult, therefore, this approach has achieved only moderate results so far.

In this paper, we propose a new principle for estimating an inverse filter by using an essential property of speech signals, namely quasi-periodicity, as a clue. In general, voiced segments in an utterance have approximate periodicity in each local time region while the period gradually changes. Therefore, when a long reverberation is added to a speech signal, signals in different time regions with different periods are mixed, thus degrading the periodicity of the signals in local time regions. By contrast, we show that we can estimate an inverse filter for dereverberating a signal by enhancing the periodicity of the signal in each

local time region. The estimated filter can dereverberate both the periodic and non-periodic parts of speech signals with no prior knowledge of the target signals, even though only the periodic parts of the signals are used for the estimation.

## 2  Quasi-periodicity based dereverberation

We propose two dereverberation methods, referred to as *Harmonicity based dEReverBeration (HERB)* methods, based on the features of quasi-periodic signals: one based on an Average Transfer Function (ATF) that transforms reverberant signals into quasi-periodic components (ATF-HERB), and the other based on the Minimum Mean Squared Error (MMSE) criterion that evaluates the quasi-periodicity of target signals (MMSE-HERB). First, we briefly explain the features of quasi-periodic signals, and then describe the two methods.

### 2.1  Features of quasi-periodic signals

When a source signal $s(n)$ is recorded in a reverberant room[1], the obtained signal $x(n)$ is represented as $x(n) = h(n) * s(n)$, where $h(n)$ is the impulse response of the room and "$*$" is a convolution operation. The goal of the dereverberation is to estimate a dereverberation filter, $w(n)$, for $-N < n < N$ that dereverberates $x(n)$, and to obtain the dereverberated signal $y(n)$ by:

$$y(n) = w(n) * x(n) = (w(n) * h(n)) * s(n) = q(n) * s(n). \qquad (1)$$

where $q(n) = w(n) * h(n)$ is referred to as a dereverberated impulse response. Here, we assume $s(n)$ is a quasi-periodic signal[2], which has the following features:

1. In each local time region around $n_0$ ($n_0 - \delta < n < n_0 + \delta$ for $^\forall n_0$), $s(n)$ is approximately a periodic signal whose period is $T(n_0)$.
2. Outside the region ($|n' - n_0| > \delta$), $s(n')$ is also a periodic signal within its neighboring time region, but often has another period that is different from $T(n_0)$.

These features make $x(n)$ a non-periodic signal even within local time regions when $h(m)$ contains non-zero values for $|m| > \delta$. This is because more than two periodic signals, $s(n)$ and $s(n-m)$, that have different periods, are added to $x(n)$ with weights of $h(0)$ and $h(m)$. Inversely, the goal of our dereverberation is to estimate $w(n)$ that makes $y(n)$ a periodic signal in each local time region. Once such a filter is obtained, $q(m)$ must have zero values for $|m| > \delta$, and thus, reverberant components longer than $\delta$ are eliminated from $y(n)$.

An important additional feature of a quasi-periodic signal is that quasi-periodic components in a source signal can be enhanced by an adaptive harmonic filter. An adaptive harmonic filter is a time-varying linear filter that enhances frequency components whose frequencies correspond to multiples of the fundamental frequency ($F_0$) of the target signal, while preserving their phases and amplitudes. The filter values are adaptively modified according to $F_0$. For example, a filter, $F(f_0(n))[\cdot]$, can be implemented as follows:

$$\hat{x}(n) = F(f_0(n))[x(n)], \qquad (2)$$

$$= \sum_{n_0} g_2(n - n_0)\text{Re}\{x(n) * (g_1(n) \sum_k \exp(j2\pi k f_0(n_0)n/f_s))\}, \qquad (3)$$

where $n_0$ is the center time of each frame, $f_0(n_0)$ is the fundamental frequency ($F_0$) of the signal at the frame, $k$ is a harmonics index, $g_1(n)$ and $g_2(n)$ are analysis window

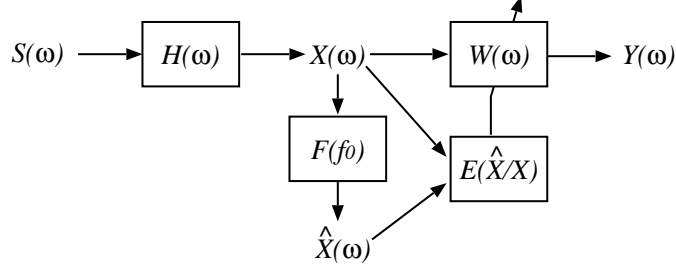

Figure 1: Diagram of ATF-HERB

functions, and $f_s$ is the sampling frequency. Even when $x(n)$ contains a long reverberation, the reverberant components that have different frequencies from $s(n)$ are reduced by the harmonic filter, and thus, the quasi-periodic components can be enhanced.

## 2.2 ATF-HERB: average transfer function based dereverberation

Figure 1 is a diagram of ATF-HERB, which uses the average transfer function from reverberant signals to quasi-periodic signals. A speech signal, $S(\omega)$, can be modeled by the sum of the quasi-periodic components, or voiced components, $S_h(\omega)$, and non-periodic components, or unvoiced components, $S_n(\omega)$, as eq. (4). The reverberant observed signal, $X(\omega)$, is then represented by the product of $S$ and the transfer function, $H(\omega)$, of a room as eq. (5). The transfer function, $H$, can also be divided into two functions, $D(\omega)$ and $R(\omega)$. The former transforms $S$ into the direct signal, $DS$, and the latter into the reverberation part, $RS$, as shown in eq. (6). $X$ is also represented by the sum of the direct signal of the quasi-periodic components, $DS_h$, and the other components as eq. (7).

$$
\begin{align}
S(\omega) &= S_h(\omega) + S_n(\omega), \tag{4}\\
X(\omega) &= H(\omega)S(\omega), \tag{5}\\
&= (D(\omega) + R(\omega))S(\omega), \tag{6}\\
&= DS_h + (RS_h + HS_n). \tag{7}
\end{align}
$$

Of these components, $DS_h$ can approximately be extracted from $X$ by harmonic filtering. Although the frequencies of quasi-periodic components change dynamically according to the changes in their fundamental frequency ($F_0$), their reverberation remains unchanged at the same frequency. Therefore, direct quasi-periodic components, $DS_h$, can be enhanced by extracting frequency components located at multiples of its $F_0$. This approximated direct signal $\hat{X}(\omega)$ can be modeled as follows:

$$\hat{X}(\omega) = D(\omega)S_h(\omega) + (\hat{R}(\omega)S_h(\omega) + \hat{N}(\omega)), \tag{8}$$

where $\hat{R}(\omega)S_h(\omega)$ and $\hat{N}(\omega)$ are part of the reverberation of $S_h$ and part of the direct signal and reverberation of $S_n$, which unexpectedly remain in $\hat{X}$ after the harmonic filtering[3]. We assume that all the estimation errors in $\hat{X}$ are caused by $\hat{R}S_h$ and $\hat{N}$ in eq. (8).

The goal of ATF-HERB is to estimate $\mathcal{O}(\hat{R}(\omega)) = (D(\omega) + \hat{R}(\omega))/H(\omega)$, referred to as a "dereverberation operator." This is because the signal $DS + \hat{R}S$, which can be obtained by multiplying $\mathcal{O}(\hat{R})$ by $X$, becomes in a sense a dereverberated signal.

$$\mathcal{O}(\hat{R}(\omega))X(\omega) = D(\omega)S(\omega) + \hat{R}(\omega)S(\omega), \tag{9}$$

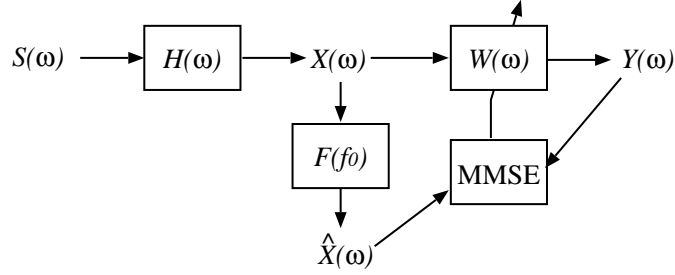

Figure 2: Diagram of MMSE-HERB

where the right side of eq. (9) is composed of a direct signal, $DS$, and certain parts of the reverberation, $\hat{R}S$. The rest of the reverberation included in $X(= DS + RS)$, or $(R - \hat{R})S$, is eliminated by the dereverberation operator.

To estimate the dereverberation operator, we use the output of the harmonic filter, $\hat{X}$. Suppose a number of $X$ values are obtained and $\hat{X}$ values are calculated from individual $X$ values. Then, the dereverberation operator, $\mathcal{O}(\hat{R})$, can be approximated as the average of $\hat{X}/X$, or $W(\omega) = E(\hat{X}/X)$. $W(\omega)$ is shown to be a good estimate of $\mathcal{O}(\hat{R})$ by substituting $E(\hat{X}/X)$ for eqs. (4), (5) and (8) as eq. (11).

$$W(\omega) = E(\hat{X}/X), \tag{10}$$

$$= \mathcal{O}(\hat{R}(\omega))E(\frac{1}{1 + S_n/S_h}) + E(\frac{1}{1 + (X - \hat{N})/\hat{N}}), \tag{11}$$

$$\simeq \mathcal{O}(\hat{R}(\omega))P(|S_h(\omega)| > |S_n(\omega)|), \tag{12}$$

where $P(\cdot)$ is a probability function. The arguments of the two average functions in eq. (11) have the form of a complex function, $f(z) = 1/(1 + z)$. $E(f(z))$ is easily proven to equal $P(|z| < 1)$, using the residue theorem if it is assumed that the phase of $z$ is uniformly distributed, the phases of $z$ and $|z|$ are independent, and $|z| \neq 1$. Based on this property, the second term of eq. (11) approximately equals zero because $\hat{N}$ is a non-periodic component that the harmonic filter unexpectedly extracts and thus the magnitude of $\hat{N}$ almost always has a smaller value than $(Y - \hat{N})$ if a sufficiently long analysis window is used. Therefore, $W(\omega)$ can be approximated by eq. (12), that is, $W(\omega)$ has the value of the dereverberation operator multiplied by the probability of the harmonic components of speech with a larger magnitude than the non-periodic components.

Once the dereverberation operator is calculated from the periodic parts of speech signals for almost all the frequency ranges, it can dereverberate both the periodic and non-periodic parts of the signals because the inverse transfer function is independent of the source signal characteristics. Instead, the gain of $W(\omega)$ tends to decrease with frequency when using our method. This is because the magnitudes of the non-periodic components relative to the periodic components tend to increase with frequency for a speech signal, and thus the $P(|S_h| > |S_n|)$ value becomes smaller as $\omega$ increases. To compensate for this decreasing gain, it may be useful to use the average attributes of speech on the probability, $P(|S_h| > |S_n|)$. In our experiments in section 4, however, $W(\omega)$ itself was used as the dereverberation operator without any compensation.

## 2.3 MMSE-HERB: minimum mean squared error criterion based dereverberation

As discussed in section 2.1, quasi-periodic signals can be dereverberated simply by enhancing their quasi-periodicity. To implement this principle directly, we introduce a cost

function, referred to as the minimum mean squared error (MMSE) criterion, to evaluate the quasi-periodicity of the signals as follows:

$$C(w) = \sum_n (y(n) - F(f_0(n))[y(n)])^2 = \sum_n (w(n) * x(n) - F(f_0(n))[w(n) * x(n)])^2,$$

(13)

where $y(n) = w(n) * x(n)$ is a target signal that should be dereverberated by controlling $w(n)$, and $F(f_0(n))[y(n)]$ is a signal obtained by applying a harmonic filter to $y(n)$. When $y(n)$ is a quasi-periodic signal, $y(n)$ approximately equals $F(f_0(n))[y(n)]$ because of the feature of quasi-periodic signals, and thus, the above cost function is expected to have the minimum value. Inversely, the filter, $w(n)$, that minimizes $C(w)$ is expected to enhance the quasi-periodicity of $x(n)$. Such filter parameters can, for example, be obtained using optimization algorithms such as a hill-climbing method using the derivatives of $C(w)$ calculated as follows:

$$\frac{\partial C(w)}{\partial w(l)} = 2 \sum_n (y(n) - F(f_0(n))[y(n)])(x(n-l) - F(f_0(n))[x(n-l)]),$$

(14)

where $F(f_0(n))[x(n-l)])$ is a signal obtained by applying the adaptive harmonic filter to $x(n-l)^4$.

There are, however, several problems involved in directly using eq. (13) as the cost function.

1. As discussed in section 2.1, the values of the dereverberated impulse response, $q(n)$, are expected to become zero using this method where $|n| > \delta$, however, the values are not specifically determined where $|n| < \delta$. This may cause unexpected spectral modification of the dereverberated signal. Additional constraints are required in order to specify these values.

2. The cost function has a self-evident solution, that is, $w(l) = 0$ for all $l$ values. This solution means that the signal, $y(n)$, is always zero instead of being dereverberated, and therefore, should be excluded. Some constraints, such as $\sum_l w(l)^2 = 1$, may be useful for solving this problem.

3. The complexity of the computing needed to minimize the cost function based on repetitive estimation increases as the dereverberation filter becomes longer. The longer the reverberation becomes, the longer the dereverberation filter should be.

To overcome these problems, we simplify the cost function in this paper. The new cost function is defined as follows:

$$C(W(\omega)) = E((Y(\omega) - \hat{X}(\omega))^2) = E((W(\omega)X(\omega) - \hat{X}(\omega))^2),$$

(15)

where $Y(\omega)$, $X(\omega)$, and $\hat{X}(\omega)$ are discrete Fourier transformations of $y(n)$, $x(n)$, and $F(f_0(n))[x(n)]$, respectively. The new cost function evaluates the quasi-periodicity not in the time domain but in the frequency domain, and uses a fixed quasi-periodic signal $\hat{X}(\omega)$ as the desired signal, instead of using the non-fixed quasi-periodic signal, $F(f_0(n))[y(n)]$. This modification allows us to solve the above problems. The use of the fixed desired signals specifically provides the dereverberated impulse response, $q(n)$, with the desired values, even in the time region, $|n| < \delta$. In addition, the self-evident solution, $w(l) = 0$, can no longer be optimal in terms of the cost function. Furthermore, the computing complexity is greatly reduced because the solution can be given analytically as follows:

$$W(\omega) = \frac{E(\hat{X}(\omega)X^*(\omega))}{E(X(\omega)X^*(\omega))}.$$

(16)

A diagram of this simplified MMSE-HERB is shown in Fig. 2.

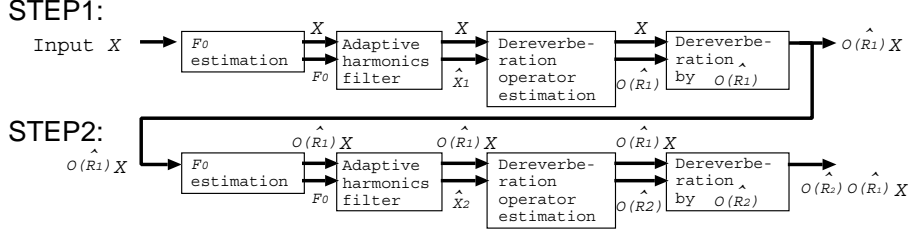

Figure 3: Processing flow of dereverberation.

When we assume the model of $\hat{X}$ in eq. (8), and $E(S_h S_n^*) = E(S_n S_h^*) = E(\hat{N} S_h^*) = 0$, it is shown that the resulting $W$ in eq. (16) again approaches the dereverberation operator, $\mathcal{O}(\hat{R})$, presented in section 2.2:

$$W(\omega) = \mathcal{O}(\hat{R}(\omega)) \frac{E(S_h S_h^*)}{E(S_h S_h^*) + E(S_n S_n^*)} + \frac{1}{H} \frac{E(\hat{N} S_n^*)}{E(S_h S_h^*) + E(S_n S_n^*)}, \quad (17)$$

$$\simeq \mathcal{O}(\hat{R}(\omega)) \frac{E(S_h S_h^*)}{E(S_h S_h^*) + E(S_n S_n^*)}. \quad (18)$$

Because $\hat{N}$ represents non-periodic components that are included unexpectedly and at random in the output of the harmonic filter, the absolute value of the second term in eq. (17) is expected to be sufficiently small compared with that of the first term, therefore, we disregard this term. Then, $W(\omega)$ in eq. (16) becomes the dereverberation operator multiplied by the ratio of the expected power of the quasi-periodic components in the signals to that of whole signals. As with the speech signals discussed in section 2.2, the $E(S_h S_h^*)/(E(S_h S_h^*) + E(S_n S_n^*))$ value becomes smaller as $\omega$ increases, and thus, the gain of $W(\omega)$ tends to decrease. Therefore, the same frequency compensation scenario as found in section 2.2 may again be useful for the MMSE based dereverberation scheme.

## 3  Processing flow

Based on the above two methods, we constructed a dereverberation algorithm composed of two steps as shown in Fig. 3. Both methods are implemented in the same processing flow except that the methods used to calculate the dereverberation operator are different. The flow is summarized as follows:

1. In the first step, $F_0$ is estimated from the reverberant signal, $X$. Then the harmonic components included in $X$ are estimated as $\hat{X}_1$ based on adaptive harmonic filtering. The dereverberation operator $\mathcal{O}(\hat{R}_1)$ is then calculated by ATF-HERB or MMSE-HERB for a number of reverberant speech signals. Finally, the dereverberated signal is obtained by multiplying $\mathcal{O}(\hat{R}_1)$ by $X$.

2. The second step employs almost the same procedures as the first step except that the speech data dereverberated by the first step are used as the input signal. The use of this dereverberated input signal means that reverberant components, $\hat{R}_2 X_2$, inevitably included in eq. (8) can be attenuated. Therefore, a more effective dereverberation can be achieved in step 2.

In our preliminary experiments, however, repeating STEP 2 did not always improve the quality of the dereverberated signals. This is because the estimation error of the dereverberation operators accumulates in the dereverberated signals when the signals are multiplied by more than one dereverberation operator. Therefore, in our experiments, we used STEP 2 only once. A more detailed explanation of these processing steps is also presented in [4].

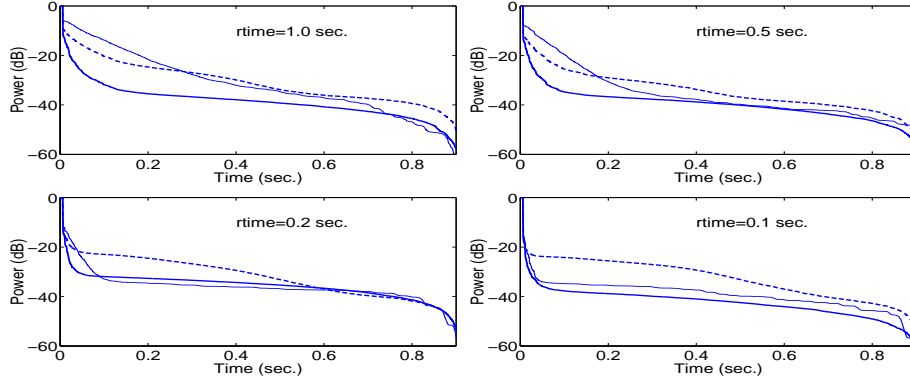

Figure 4: Reverberation curves of the original impulse responses (thin line) and dereverberated impulse responses (male: thick dashed line, female: thick solid line) for different reverberation times (rtime).

Accurate $F_0$ estimation is very important in terms of achieving effective dereverberation with our methods in this processing flow. However, this is a difficult task, especially for speech with a long reverberation using existing $F_0$ estimators. To cope with this problem, we designed a simple filter that attenuates a signal that continues at the same frequency, and used it as a preprocessor for the $F_0$ estimation [5]. In addition, the dereverberation operator, $\mathcal{O}(\hat{R}_1)$, itself is a very effective preprocessor for an $F_0$ estimator because the reverberation of the speech can be directly reduced by the operator. This mechanism is already included in step 2 of the dereverberation procedure, that is, $F_0$ estimation is applied to $\mathcal{O}(\hat{R}_1)X$. Therefore, more accurate $F_0$ can be obtained in step 2 than in step 1.

## 4   Experimental results

We examined the performance of the proposed dereverberation methods. Almost the same results were obtained with the two methods, and so we only describe those obtained with ATF-HERB. We used 5240 Japanese word utterances provided by a male and a female speaker (MAU and FKM, 12 kHz sampling) included in the ATR database as source signals, $S(\omega)$. We used four impulse responses measured in a reverberant room whose reverberation times were about 0.1, 0.2, 0.5, and 1.0 sec, respectively. Reverberant signals, $X(\omega)$, were obtained by convolving $S(\omega)$ with the impulse responses.

Figure 4 depicts the reverberation curves[5] of the original impulse responses and the dereverberated impulse responses obtained with ATF-HERB. The figure shows that the proposed methods could effectively reduce the reverberation in the impulse responses for the female speaker when the reverberation time (rtime) was longer than 0.1 sec. For the male speaker, the reverberation effect in the lower time region was also effectively reduced. This means that strong reverberant components were eliminated, and we can expect the intelligibility of the signals to be improved [6].

Figure 5 shows spectrograms of reverberant and dereverberated speech signals when rtime was 1.0 sec. As shown in the figure, the reverberation of the signal was effectively reduced, and the formant structure of the signal was restored. Similar spectrogram features were observed under other reverberation conditions, and an improvement in sound quality could clearly be recognized by listening to the dereverberated signals [7]. We also evaluated the quality of the dereverberated speech in terms of speaker dependent word recognition rates

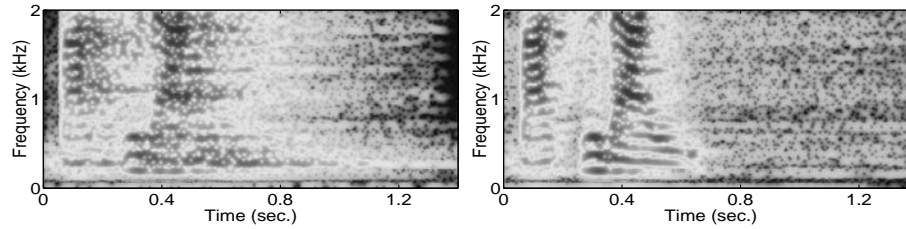

Figure 5: Spectrogram of reverberant (left) and dereverberated (right) speech of a male speaker uttering "ba-ku-da-i".

with an ASR system, and could achieve more than 95 % recognition rates under all the reverberation conditions with acoustic models trained using dereverberated speech signals. Detailed information on the ASR experiments is also provided in [4].

## 5   Conclusion

A new blind dereverberation principle based on the quasi-periodicity of speech signals was proposed. We presented two types of dereverberation method, referred to as harmonicity based dereverberation (HERB) method: one estimates the average filter function that transforms reverberant signals into quasi-periodic signals (ATF-HERB) and the other minimizes the MMSE criterion that evaluates the quasi-periodicity of signals (MMSE-HERB). We showed that ATF-HERB and a simplified version of MMSE-HERB are both capable of learning the dereverberation operator that can reduce reverberant components in speech signals. Experimental results showed that a dereverberation operator trained with 5240 Japanese word utterances could achieve very high quality speech dereverberation. Future work will include an investigation of how such high quality speech dereverberation can be achieved with fewer speech data.

## Footnotes

[1]In this paper, time domain and frequency domain signals are represented by non-capitalized and capitalized symbols, respectively. Arguments "($\omega$)" that represent the center frequencies of the discrete Fourier transformation bins are often omitted from frequency domain signals.

[2]Later, this assumption is extended so that $s(n)$ is composed of quasi-periodic components and non-periodic components in the case of speech signals.

[3]Strictly speaking, $\hat{R}$ cannot be represented as a linear transformation because the reverberation included in $\hat{X}$ depends on the time pattern of $\hat{X}$. We introduce this approximation for simplicity.

[4] $F(f_0(n))[x(n-l)])$ is not the same signal as $\hat{x}(n-l)$. When calculating $F(f_0(n))[x(n-l)]$, $x(n)$ is time-shifted with $l$-points while $f_0(n)$ of the adaptive harmonic filter is not time-shifted.

[5]The reverberation curve shows the reduction in the energy of a room impulse response with time [6].

## References

[1] Baba, A., Lee, A., Saruwatari, H., and Shikano, K., "Speech recognition by reverberation adapted acoustic model," *Proc. of ASJ general meeting*, pp. 27–28, Akita, Japan, Sep., 2002.

[2] Amari, S., Douglas, S. C., Cichocki, A., and Yang, H. H., "Multichannel blind deconvolution and equalization using the natural gradient," *Proc. IEEE Workshop on Signal Processing Advances in Wireless Communications*, Paris, pp. 101-104, April 1997.

[3] Yegnanarayana, B., and Murthy, P. S., "Enhancement of reverberant speech using LP residual signal," *IEEE Trans. SAP* vol. 8, no. 3, pp. 267–281, 2000.

[4] Nakatani, T., Miyoshi, M., and Kinoshita, K., "Implementation and effects of single channel dereverberation based on the harmonic structure of speech," *Proc. IWAENC-2003*, Sep., 2003.

[5] Nakatani, T., and Miyoshi, M., "Blind dereverberation of single channel speech signal based on harmonic structure," *Proc. ICASSP-2003*, vol. 1, pp. 92–95, Apr., 2003.

[6] Yegnanarayana, B., and Ramakrishna, B. S., "Intelligibility of speech under nonexponential decay conditions," *JASA*, vol. 58, pp. 853–857, Oct. 1975.

[7] http://www.kecl.ntt.co.jp/icl/signal/nakatani/sound-demos/dm/derev-demos.html
